# The Neural Costs of Optimal Control

**Samuel J. Gershman and Robert C. Wilson**
Psychology Department and Neuroscience Institute
Princeton University
Princeton, NJ 08540
`{sjgershm,rcw2}@princeton.edu`

## Abstract

Optimal control entails combining probabilities and utilities. However, for most practical problems, probability densities can be represented only approximately. Choosing an approximation requires balancing the benefits of an accurate approximation against the costs of computing it. We propose a variational framework for achieving this balance and apply it to the problem of how a neural population code should optimally represent a distribution under resource constraints. The essence of our analysis is the conjecture that population codes are organized to maximize a lower bound on the log expected utility. This theory can account for a plethora of experimental data, including the reward-modulation of sensory receptive fields, GABAergic effects on saccadic movements, and risk aversion in decisions under uncertainty.

## 1   Introduction

Acting optimally under uncertainty requires comparing the expected utility of each possible action, but in most situations of practical interest this expectation is impossible to calculate exactly: the hidden states that must be integrated over may be high-dimensional and the probability density may not take on any simple form. As a consequence, approximations must inevitably be used. Typically one has a choice of approximation, with more exact approximations demanding more computational resources, a penalty that can be naturally incorporated into the utility function. The question we address in this paper is: given a family of approximations and their associated resource demands, what approximation will lead as close as possible to the optimal control policy?

This is a poignant problem for the brain, which expends a collosal amount of metabolic energy in building an internal model of the world. Previous theoretical work has studied how "energy-efficient codes" might be constructed by the brain to maximize information transfer with the least possible energy consumption [10]. However, maximizing information transfer is only one component of adaptive behavior; the utility of information must be taken into account when choosing a code [15], and this may interact in complicated ways with the computational costs of approximate inference.

Our contribution is to place this problem within a decision-theoretic framework by representing the choice of approximation as a "meta-decision" with its own expected utility. Central to our analysis is the observation that while this expected utility cannot be maximized directly, it is possible to maximize a variational lower bound on log expected utility (see also [17, 5] for related approaches). We study the properties of this lower bound and show how it accounts for some intriguing empirical properties of neural codes.

## 2  Optimal control with approximate densities

Let $a$ denote an action and $s$ denote a hidden state variable drawn from some probability density $p(s)$.[1] Given a utility function $U(a; s)$, the optimal action $a_p$ is the one that maximizes expected utility $V_p(a)$:

$$a_p = \operatorname*{argmax}_a V_p(a), \tag{1}$$

where

$$V_p(a) = \mathbb{E}_p[U(a; s)] = \int_s p(s)U(a; s)ds. \tag{2}$$

Computing the expected utility for each action requires solving a possibly intractable integral. An approximation of expected utility can be obtained by substituting an alternative density $q(s)$ for which the expected utility is tractable. For example, one might choose $q(s)$ to be a Gaussian with some mean and variance, or a Monte Carlo approximation, or even a delta function at some point.

Using an approximate density presents the "meta-decision" of which density to use. If one chooses optimally under $q(s)$, then the expected utility is given by $\mathbb{E}_p[U(a_q; s)] = V_p(a_q)$, therefore the optimal density $q^*$ should be chosen according to

$$q^* = \operatorname*{argmax}_{q \in \mathcal{Q}} V_p(a_q), \tag{3}$$

where $\mathcal{Q}$ is some family of densities. To understand Eq. 3, consider the optimization as consisting of two parts: first, select an approximate density $q(s)$ and choose the optimal action with respect to this density; then evaluate the *true* value of that action under the target density. Clearly, if $p \in \mathcal{Q}$, then $q = p$ is the optimal solution. In general, we cannot optimize this function directly because it requires solving precisely the integral we are trying to avoid: the expected utility under $p(s)$. We can, however, use the approximate density to lower-bound the log expected utility under $p(s)$ by appealing to Jensen's inequality:

$$\log V_p(a) \geq \int_s q(s) \log \frac{p(s)U(a; s)}{q(s)}ds$$
$$= \mathbb{E}_q[\log U(a; s)] + \mathbb{E}_q[\log p(s)] - \mathbb{E}_q[\log q(s)], \tag{4}$$

Notice the similarity to the evidence lower bound used in variational Bayesian inference [9]: whereas in variational inference we attempt to lower-bound the log marginal likelihood (evidence), in variational decision theory we attempt to lower-bound the log expected utility.

Examining the utility lower bound, we see that the terms exert conceptually distinct influences:

1. A *utility* component, $\mathbb{E}_q[\log U(a; s)]$, the expected log utility under the approximate density.

2. A *cross-entropy* component, $-\mathbb{E}_q[\log p(s)]$, reflecting the mismatch between the approximate density and the target density. This can be thought of as a form of "sensory prediction error."

3. An *entropy* component, $-\mathbb{E}_q[\log q(s)]$, embodying a *maximum entropy principle* [8]: for a fixed utility and cross-entropy, choose the distribution with maximal entropy.

Intuitively, a more accurate approximate density $q(s)$ should incur a larger computational cost. One way to express this notion of cost is to incorporate it directly into the utility function. That is, we consider an augmented utility function $U(a, q; s)$ that depends on the approximate density. If we assume that the utility function takes the form $\log U(a, q; s) = \log R(a; s) - \log C(q)$, where $R(a; s)$ represents a reward function and $C(q)$ represents a computational cost function, we arrive at the following modification to the utility lower bound:

$$\mathcal{L}(q, a) = \mathbb{E}_q[\log R(a; s)] + \mathbb{E}_q[\log p(s)] - \mathbb{E}_q[\log q(s)] - \log C(q). \tag{5}$$

The assumption that the log utility decomposes into additive reward and cost components is intuitive: it implies that reward is measured relative to the computational cost of earning it. In summary, the utility lower bound $\mathcal{L}(q, a)$ provides an objective function for simultaneously choosing an action and choosing an approximate density over hidden states. Whereas in classical decision theory, optimization is performed over the action space, in variational decision theory optimization is performed over the joint space of actions and approximate densities. Perception and action are thereby treated as a single optimization problem.

## 3 Choosing a probabilistic population code

While the theory developed in the previous section applies to any representation scheme, in this section, for illustrative purposes, we focus on one specific family of approximate densities defined by the firing rate of neurons in a network. Specifically, we consider a population of $N$ neurons tasked with encoding a probability density over $s$. One way to do this, known as a kernel density estimate (KDE) code [1, 28], is to associate with each neuron a kernel density $f_n(s)$ and then approximate the target density with a convex combination of the kernel densities:

$$q(s) = \frac{1}{Z} \sum_{n=1}^{N} e^{x_n} f_n(s), \tag{6}$$

where $x_n$ denotes the firing rate of neuron $n$ and $Z = \sum_{n=1}^{N} e^{x_n}$. We assume that the kernel density functions are Gaussian, parameterized by a preferred stimulus (mean) $s_n$ and a standard deviation $\sigma_n$:

$$f_n(s) = \frac{1}{\sqrt{2\pi}\sigma_n} \exp\left(-\frac{(s - s_n)^2}{2\sigma_n^2}\right) \tag{7}$$

For simplicity, in this paper we will focus on the limiting case in which $\sigma \Rightarrow 0$.[2] In this case $q(s)$ degenerates onto a collection of delta functions:

$$q(s) = \frac{1}{Z} \sum_{n=1}^{N} e^{x_n} \delta(s - s_n), \tag{8}$$

where $\delta(\cdot)$ is the Dirac delta function. This density corresponds to a collection of sharply tuned neurons; provided that the preferred values $\{s_1, \ldots, s_N\}$ densely cover the state space, $q(s)$ can represent arbitrarily complicated densities by varying the firing rates $\mathbf{x}$.

### 3.1 Optimizing the bound

Assuming for the moment that there is only a single action, we can state the optimization problem as follows: given the family of approximate densities parameterized by $\mathbf{x}$, choose the density that maximizes the utility lower bound

$$\mathcal{L}(q, a) = \frac{1}{Z} \sum_{n=1}^{N} e^{x_n} \left[\log U(a; s_n) + \log \tilde{p}(s_n) - x_n\right] + \log Z - \log \mathcal{B} - \log C(q), \tag{9}$$

where $p(s) = \tilde{p}(s)/\mathcal{B}$ (i.e., $\tilde{p}(s)$ is the un-normalized target density). Note also that $\mathcal{B} = \int_s \tilde{p}(s)ds$ does not depend on $x_n$, and hence can be ignored for the purposes of optimization. Technically, the lower bound is not well defined in the limit because the target density is non-atomic (i.e., has zero mass at any given value). However, approximating the expectations in Eq. 5 by $\mathbb{E}_q[g(s)] \approx Z^{-1} \sum_{n=1}^{N} e^{x_n} g(s_n)$, as we do above, can be justified in terms of first-order Taylor series expansions around the preferred stimuli, which will be arbitrarily accurate as $\sigma \to 0$.

In the rest of this paper, we shall assume that the cost function takes the following form:

$$C(q) = \beta N + \gamma \sum_{n=1}^{N} x_n, \tag{10}$$

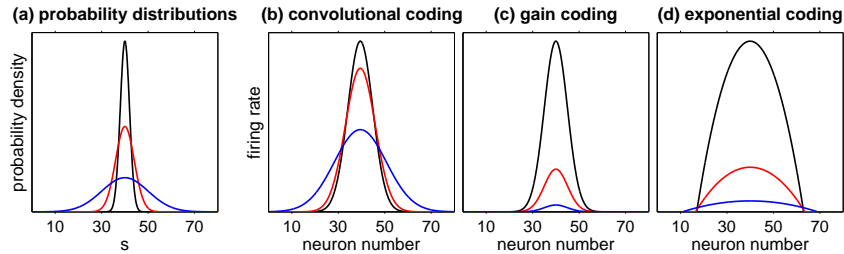

Figure 1: **Comparison between coding schemes**. The leftmost panel shows a collection of probability distributions with different variances, and the other panels show different neural representations of these distributions.

where $\beta$ is the fixed cost of maintaining a neuron, and $\gamma$ is the cost of a spike (c.f. [10]).

We next seek a neuronal update rule that performs gradient ascent on the utility lower bound. Holding the firing rate of all neurons except $n$ fixed, taking the partial derivative of $\mathcal{L}(q, a)$ with respect to $x_n$ and setting it to 0, we arrive at the following update rule:

$$
x_n \leftarrow \left[ \log U(a; s_n) + \log \tilde{p}(s_n) + \frac{1}{Z} \sum_{j=1}^{N} e^{x_j} \left[ x_j - \log U(a; s_j) - \log \tilde{p}(s_j) \right] - \frac{Z\gamma}{e^{x_n} C(q)} \right]_+
$$
(11)

where $[\cdot]_+$ denotes linear rectification.[3] This update rule defines an attractor network whose Lyapunov function is the (negative) utility lower bound. When multiple actions are involved, the bound can be jointly optimized over $a$ and $q$ by coordinate ascent. While somewhat untraditional, we note that this update rule is biologically plausible in the sense that it only involves local pairwise interactions between neurons.

## 4    Relation to other probability coding schemes

### 4.1    Exponential, convolutional and gain coding

The probability coding scheme proposed in Eq. 8 is closely related to the exponential coding described in [16]. That scheme also encodes probabilities using exponentiated activities, although it uses the representation in a very different way and in a network with very different dynamics, focusing on sequential inference problems instead of the arbitrary decision problems we consider here. Other related schemes include convolutional coding [28], in which a distribution is encoded by convolving it with a neural tuning function, and gain coding [11, 27], in which the variance of the distribution is inversely proportional to the gain of the neural response.

In Figure 1, we show how these three different ways of encoding probability distributions represent three different Gaussians with variance 2 (black line in Figure 1a), 4 (red) and 10 (blue) units. Convolutional coding (Figure 1b) is characterized by a neural response pattern that gets broader as the distribution gets broader. This has been one of the major criticisms of this type of encoding scheme as this result does not seem to be borne out experimentally (e.g., [19, 2]). In contrast, gain coding schemes (Figure 1c) posit that changes in uncertainty only change the overall gain, and not the shape, of the neural response. This leads to predictions that are consistent with experiments, but limits the type of distributions that can be represented to the exponential family [11].

Finally, Figure 1d shows how the exponential coding scheme we propose represents the distributions in a manner that can be thought of as in between convolutional coding and gain encoding, with a population response that gets broader as the encoded distribution broadens, but in a much less

pronounced way than pure convolutional coding. This point is crucial for the biological plausibility of this scheme, as it seems unlikely that these minute differences in population response width would be easily measured experimentally.

It is also important to note that both the convolutional and gain coding schemes ignore the utility function in constructing probabilistic representations. As we explore in later sections, rewards and costs place strong constraints on the types of codes that are learned by the variational objective, and the available experimental data is congruent with this view. "Pure" probabilistic representations may not exist in the brain.

## 4.2 Connection to Monte Carlo approximation

Substantial interest has been generated recently in the idea that the brain might use some form of sampling (i.e., Monte Carlo algorithm) to approximate complicated probability densities. Psychological phenomena like perceptual multistability [6] and speech perception [21] are parsimoniously explained by a model in which a density over the complete hypothesis space is replaced by a small set of discrete samples. Thus, it is reasonable to speculate whether our theory of population coding relates to these at the neural level.

When each neuron's tuning curve is sharply peaked, the resulting population code resembles *importance sampling*, a common Monte Carlo method for approximating probability densities, wherein the approximation consists of a weighted set of samples:

$$p(s) \approx \sum_{n=1}^{N} w^{(n)} \delta(s - s^{(n)}), \tag{12}$$

where $s^{(n)}$ is drawn from a proposal density $\pi(s)$ and $w^{(n)} \propto p(s^{(n)})/\pi(s^{(n)})$. In fact, we can make this correspondence precise: for any population code of the form in Eq. 8, there exists an equivalent importance sampling approximation. The corresponding proposal density takes the form:

$$\pi(s) \propto \sum_{n} \frac{p(s_n)}{e^{x_n}} \delta(s - s_n). \tag{13}$$

This means that optimizing the bound with respect to $\mathbf{x}$ is equivalent to selecting a proposal density so as to maximize utility under resource constraints. A related analysis was made by Vul et al. [26], though in a more restricted setting, showing that maximal utility is achieved with very few samples when sampling is costly. Similarly, $\pi(s)$ will be sensitive to the computational costs inherent in the utility lower bound, favoring a small number of samples.

Interestingly, importance sampling has been proposed as a neurally-plausible mechanism for Bayesian inference [22]. In that treatment, the proposal density was assumed to be the prior, leading to the prediction that neurons with preferred stimulus $s^*$ should occur with frequency proportional to the prior probability of $s^*$. One source of evidence for this prediction comes from the *oblique effect*: the observation that more V1 neurons are tuned to cardinal orientations than to oblique orientations [3], consistent with the statistics of the natural visual environment. In contrast, our model predicts that the proposal density will be sensitive to rewards in addition to the prior; as we argue in the section 5.1, a considerable amount of evidence favors this view.

## 5 Results

In the following sections, we examine some of the neurophysiological and psychological implications of the variational objective. Tying these diverse topics together is the central idea that utilities, costs and probabilistic beliefs exert a synergistic effect on neural codes and their behavioral outputs. One consequence of the variational objective is that a clear separation of these components in the brain may not exist: rewards and costs infiltrate very early sensory areas. These influences result in distortions of probabilistic belief that appear robustly in experiments with humans and animals.

## 5.1 Why are sensory receptive fields reward-modulated?

Accumulating evidence indicates that perceptual representations in the brain are modulated by reward expectation. For example, Shuler and Bear [23] paired retinal stimulation of the left and right

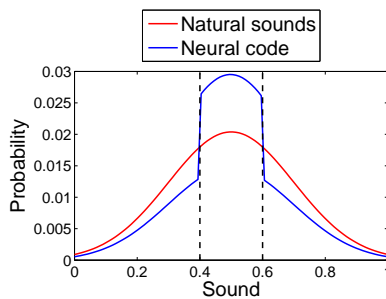

Figure 2: **Grasshopper auditory coding**. Probability density of natural sounds and the optimized approximate density, with black lines demarcating the region of behaviorally relevant sounds.

eyes with reward after different delays and recorded neurons in primary visual cortex that switched from representing purely physical attributes of the stimulation (e.g., eye of origin) to coding reward timing. Similarly, Serences [20] showed that spatially selective regions of visual cortex are biased by the prior reward associated with different spatial locations. These studies raise the possibility that the brain does not encode probabilistic beliefs separately from reward; indeed, this idea has been enshrined by a recent theoretical account [4]. One important ramification of this conflation is that it would appear to violate one of the axioms of statistical decision theory: *probabilistic sophistication* [18]. On the other hand, the variational framework we have described accounts for these findings by showing that decision-making using approximate densities leads automatically to reward-modulated probabilistic beliefs. Thus, the apparent inconsistency with statistical decision theory may be an artifact of rational responses to the information-processing constraints of the brain.

To drive this point home, we now analyze one example in more detail. Machens et al. [12] recorded the responses of grasshopper auditory neurons to different stimulus ensembles and found that the ensembles that elicited the optimal response differed systematically from the natural auditory statistics of the grasshopper's environment. In particular, the optimal ensembles were restricted to a region of stimulus space in which behaviorally important sounds live, namely species-specific mating signals. In the words of Machens et al., "an organism may seek to distribute its sensory resources according to the behavioral relevance of the natural stimuli, rather than according to purely statistical principles." We modeled this phenomenon by constructing a relatively wide density of natural sounds with a narrow region of behaviorally relevant sounds (in which states are twice as rewarding). Figure 2 shows the results, confirming that maximizing the utility lower bound selects a kernel density estimate that is narrower than the target density of natural sounds.

### 5.2 Changing the cost of a spike

Experimentally, there are at least two ways to manipulate the cost of a spike. One is by changing the amount of inhibition in the network (e.g., using injections of muscimol, a GABA agonist) and hence increasing the metabolic requirements for action potential generation. A second method is by manipulating the availability of glucose [7], either by making the subject hypoglycemic or by administering local infusions of glucose directly into the brain. We predict that increasing spiking costs (either by reducing glucose levels or increasing GABAergic transmission) will result in a diminished ability to detect weak signals embedded in noise. Consistent with this prediction, controlled hypoglycemia reduces the speed with which visual changes are detected amidst distractors [13].

These predictions have received a more direct test in a recent visual search experiment by McPeek and Keller [14], in which muscimol was injected into local regions of the superior colliculus, a brain area known to control saccadic target selection. In the absence of distractors, response latencies to the target were increased when it appeared in the receptive fields of the inhibited neurons. In the presence of distractors, response latencies increased and choice accuracy decreased when the target appeared in the receptive fields of the inhibited neurons. We simulated these findings by constructing a cost-field $\gamma(n)$ to represent the amount of GABAergic transmission at different neurons induced by muscimol injections. In the distractor condition (Figure 3, top panel), accuracy

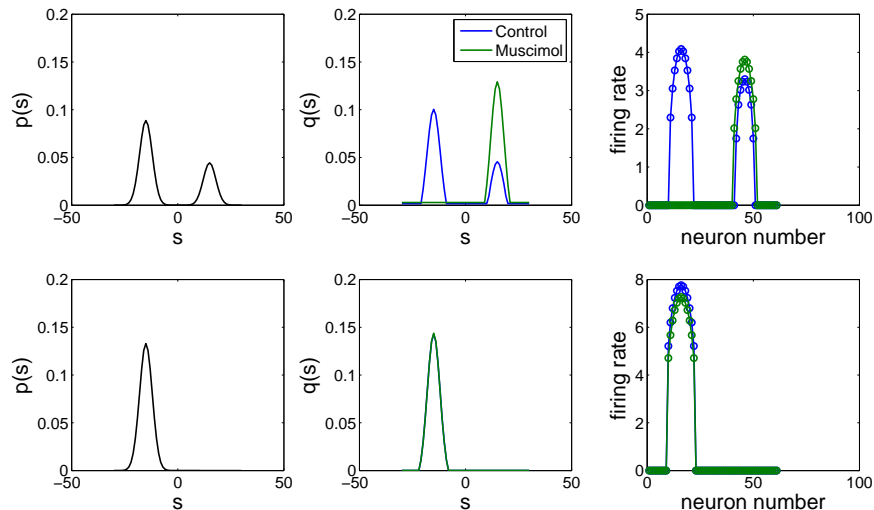

Figure 3: **Spiking cost in the superior colliculus**. Top panels illustrate distractor condition. Bottom panels illustrate no-distractor condition. (*Left column*) Target density, with larger bump in the top panel representing the target; (*Center column*) neural code under different settings of cost-field $\gamma(n)$; (*Right column*) firing rates under different cost-fields.

decreases because the increased cost of spiking in the neurons representing the target location dampens the probability density in that location. Increasing spiking cost also reduces the overall firing rate in the target-representing neurons relative to the distractor-representing neurons. This predicts increased response latencies if we assume a monotonic relationship with the relative firing rate in the target-representing neurons. Similarly, in the no-distractor condition (Figure 3, bottom panel), response latencies increase due to decreased firing rate in the target-representing neurons.

### 5.3 Non-linear probability weighting

In this section, we show that the variational objective provides a new perspective on some well-known peculiarities of human probabilistic judgment. In particular, the ostensibly irrational non-linear weighting of probabilities in risky choice emerges naturally from optimization of the variational objective under a natural assumption about the ecological distribution of rewards.

Tversky and Kahneman [25] observed that people tend to be risk-seeking (over-weighting probabilities) for low-probability gains and risk-averse (under-weighting probabilities) for high-probability gains. This pattern reverses for losses. The variational objective explains these phenomena by virtue of the fact that under neural resource constraints, the approximate density will be biased towards high reward regions of the state space. It is also necessary to assume that the magnitude of gains or losses scales inversely with probability (i.e., large gains or losses are rare). With this assumption, the optimized neural code produce the four-fold pattern of risk-attitudes observed by Tversky and Kahneman (Figure 4).

## 6 Discussion

We have presented a variational objective function for neural codes that balances motivational, statistical and metabolic demands in the service of optimal behavior. The essential idea is that the intractable problem of computing expected utilities can be finessed by instead computing expected utilities under an approximate density that optimizes a variational lower bound on log expected utility. This lower bound captures the neural costs of optimal control: more accurate approximations will require more metabolic resources, whereas less accurate approximations will diminish the amount of earned reward. This principle can explain, among other things, why receptive fields of

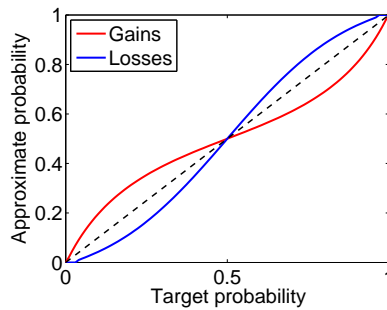

Figure 4: **Probability weighting**. Simulated calibration curve for gains and losses. Perfect calibration (i.e., linear weighting) is indicated by the dashed line.

sensory neurons have repeatedly been found to be sensitive to reward contingencies. Intuitively, expending more resources on accurately approximating the complete density of natural sensory statistics is inefficient (from an optimal control perspective) if the behaviorally relevant signals live in a compact subspace. We showed that the approximation that maximizes the utility lower bound concentrates its density within this subspace.

Our variational framework differs in important ways from the one recently proposed by Friston [4]. In his treatment, utilities are not represented explicitly at all; rather, they are implicit in the probabilistic structure of the environment. Based on an evolutionary argument, Friston suggests that high utility states are precisely those that have high probability, since otherwise organisms who find themselves frequently in low utility states are unlikely to survive. Thus, adopting a control policy that minimizes a variational upper bound on surprise will lead to optimal behavior. However, adopting this control policy may lead to pathological behaviors, such as attraction to malign states that have been experienced frequently (e.g., a person who has been poor her whole life should reject a winning lottery ticket). In contrast, our variational framework is motivated by quite different considerations arising from the computational constraints of the brain's architecture. Nonetheless, these approaches have in common the idea that probabilistic beliefs will be shaped by the utility structure of the environment.

The psychological concept of "bounded rationality" is an old one [24], classically associated with the observation that humans sometimes adopt strategies for identifying adequate solutions rather than optimal ones ("satisficing"). The variational framework offers a rather different perspective on bounded rationality; it asserts that humans are indeed trying to find optimal solutions, but subject to certain computational resource constraints. By making explicit what these constraints are, and how they interact at a neural level, our work provides a foundation upon which to develop a more complete neurobiological theory of optimal control under resource constraints.

### Acknowledgments

We thank Matt Botvinick, Matt Hoffman, Chong Wang, Nathaniel Daw and Yael Niv for helpful discussions. SJG was supported by a Quantitative Computational Neuroscience grant from the National Institutes of Health.

## Footnotes

[1]For the sake of notational simplicity, we implicitly condition on any observed variables. We also refer throughout this paper to probability densities over a multimdensional, continuous state variable, but our results still apply to one dimensional and discrete variables (in which case the probability densities are replaced with probability mass functions).

[2]The case of small, finite $\sigma$ can be addressed by using a Laplace approximation to the integrals and leads to small correction terms in the following equations.

[3]This update is equivalent to performing gradient ascent on $\mathcal{L}$ with a variable learning rate parameter given by $\frac{Z}{e^{x_n}}$. We chose this rule as it converges faster and seems more neurally plausible than the pure gradient ascent.

## References

[1] C.H. Anderson and D.C. Van Essen. Neurobiological computational systems. *Computational intelligence imitating life*, pages 213–222, 1994.

[2] J.S. Anderson, I. Lampl, D.C. Gillespie, and D. Ferster. The contribution of noise to contrast invariance of orientation tuning in cat visual cortex. *Science*, 290(5498):1968, 2000.

[3] R.L. De Valois, E. William Yund, and N. Hepler. The orientation and direction selectivity of cells in macaque visual cortex. *Vision Research*, 22(5):531–544, 1982.

[4] K. Friston. The free-energy principle: a unified brain theory? *Nature Reviews Neuroscience*, 11(2):127–138, 2010.

[5] T. Furmston and D. Barber. Variational methods for reinforcement learning. *Proceedings of the Thirteenth Conference on Artificial Intelligence and Statistics (AISTATS)*, 2010.

[6] S.J. Gershman, E. Vul, and J.B. Tenenbaum. Perceptual multistability as Markov Chain Monte Carlo inference. In Y. Bengio, D. Schuurmans, J. Lafferty, C. K. I. Williams, and A. Culotta, editors, *Advances in Neural Information Processing Systems 22*, pages 611–619. 2009.

[7] P.E. Gold. Role of glucose in regulating the brain and cognition. *American Journal of Clinical Nutrition*, 61:987S–995S, 1995.

[8] E.T. Jaynes. On the rationale of maximum-entropy methods. *Proceedings of the IEEE*, 70(9):939–952, 1982.

[9] M.I. Jordan, Z. Ghahramani, T.S. Jaakkola, and L.K. Saul. An introduction to variational methods for graphical models. *Machine learning*, 37(2):183–233, 1999.

[10] W.B. Levy and R.A. Baxter. Energy efficient neural codes. *Neural Computation*, 8(3):531–543, 1996.

[11] W.J. Ma, J.M. Beck, P.E. Latham, and A. Pouget. Bayesian inference with probabilistic population codes. *Nature Neuroscience*, 9(11):1432–1438, 2006.

[12] C.K. Machens, T. Gollisch, O. Kolesnikova, and A.V.M. Herz. Testing the efficiency of sensory coding with optimal stimulus ensembles. *Neuron*, 47(3):447–456, 2005.

[13] RJ McCrimmon, IJ Deary, BJP Huntly, KJ MacLeod, and BM Frier. Visual information processing during controlled hypoglycaemia in humans. *Brain*, 119(4):1277, 1996.

[14] R.M. McPeek and E.L. Keller. Deficits in saccade target selection after inactivation of superior colliculus. *Nature neuroscience*, 7(7):757–763, 2004.

[15] P.R. Montague and B. King-Casas. Efficient statistics, common currencies and the problem of reward-harvesting. *Trends in cognitive sciences*, 11(12):514–519, 2007.

[16] R.P.N. Rao. Bayesian computation in recurrent neural circuits. *Neural Computation*, 16(1):1–38, 2004.

[17] M. Sahani. A biologically plausible algorithm for reinforcement-shaped representational learning. *Advances in Neural Information Processing*, 16, 2004.

[18] L.J. Savage. *The Foundations of Statistics*. Dover, 1972.

[19] G. Sclar and RD Freeman. Orientation selectivity in the cat's striate cortex is invariant with stimulus contrast. *Experimental Brain Research*, 46(3):457–461, 1982.

[20] J.T. Serences. Value-based modulations in human visual cortex. *Neuron*, 60(6):1169–1181, 2008.

[21] L. Shi, N.H. Feldman, and T.L. Griffiths. Performing Bayesian inference with exemplar models. In *Proceedings of the 30th annual conference of the cognitive science society*, pages 745–750, 2008.

[22] Lei Shi and Thomas Griffiths. Neural implementation of hierarchical bayesian inference by importance sampling. In Y. Bengio, D. Schuurmans, J. Lafferty, C. K. I. Williams, and A. Culotta, editors, *Advances in Neural Information Processing Systems 22*, pages 1669–1677. 2009.

[23] M.G. Shuler and M.F. Bear. Reward timing in the primary visual cortex. *Science*, 311(5767):1606, 2006.

[24] H.A. Simon. *Models of Bounded Rationality*. MIT Press, 1982.

[25] A. Tversky and D. Kahneman. Advances in prospect theory: cumulative representation of uncertainty. *Journal of Risk and uncertainty*, 5(4):297–323, 1992.

[26] E. Vul, N.D. Goodman, T.L. Griffiths, and J.B. Tenenbaum. One and done? Optimal decisions from very few samples. In *Proceedings of the 31st Annual Meeting of the Cognitive Science Society, Amsterdam, the Netherlands*, 2009.

[27] R.C. Wilson and L.H. Finkel. A neural implementation of the kalman filter. In Y. Bengio, D. Schuurmans, J. Lafferty, C. K. I. Williams, and A. Culotta, editors, *Advances in Neural Information Processing Systems 22*, pages 2062–2070. 2009.

[28] R.S. Zemel, P. Dayan, and A. Pouget. Probabilistic interpretation of population codes. *Neural Computation*, 10(2):403–430, 1998.

